# Speeding up the Parti-Game Algorithm

**Maxim Likhachev**
School of Computer Science
Carnegie Mellon University
Pittsburgh, PA 15213
maxim+@cs.cmu.edu

**Sven Koenig**
College of Computing
Georgia Institute of Technology
Atlanta, GA 30312-0280
skoenig@cc.gatech.edu

## Abstract

In this paper, we introduce an efficient replanning algorithm for nondeterministic domains, namely what we believe to be the first incremental heuristic minimax search algorithm. We apply it to the dynamic discretization of continuous domains, resulting in an efficient implementation of the parti-game reinforcement-learning algorithm for control in high-dimensional domains.

## 1   Introduction

We recently developed Lifelong Planning A* (LPA*), a search algorithm for deterministic domains that combines incremental and heuristic search to reduce its search time [1]. Incremental search reuses information from previous searches to find solutions to series of similar search tasks faster than is possible by solving each search task from scratch [2], while heuristic search uses distance estimates to focus the search and solve search problems faster than uninformed search. In this paper, we extend LPA* to nondeterministic domains. We believe that the resulting search algorithm, called Minimax LPA*, is the first incremental heuristic minimax search algorithm. We apply it to the dynamic discretization of continuous domains, resulting in an efficient implementation of the popular parti-game algorithm [3]. Our first experiments suggest that this implementation of the parti-game algorithm can be an order of magnitude faster in two-dimensional domains than one with uninformed search from scratch and thus might allow the parti-game algorithm to scale up to larger domains. There also exist other ways of decreasing the amount of search performed by the parti-game algorithm. We demonstrate some advantages of Minimax LPA* over Prioritized Sweeping [4] in [5] but it is future work to compare it with the algorithms developed in [6].

## 2   Parti-Game Algorithm

The objective of the parti-game algorithm is to move an agent from given start coordinates to given goal coordinates in continuous and potentially high-dimensional domains with obstacles of arbitrary shapes. It is popular because it is simple, efficient, and applies to a broad range of control problems. To solve these problems, one can first discretize the domains and then use conventional search algorithms to determine plans that move the agent to the goal coordinates. However, uniform discretizations can prevent one from finding a plan if

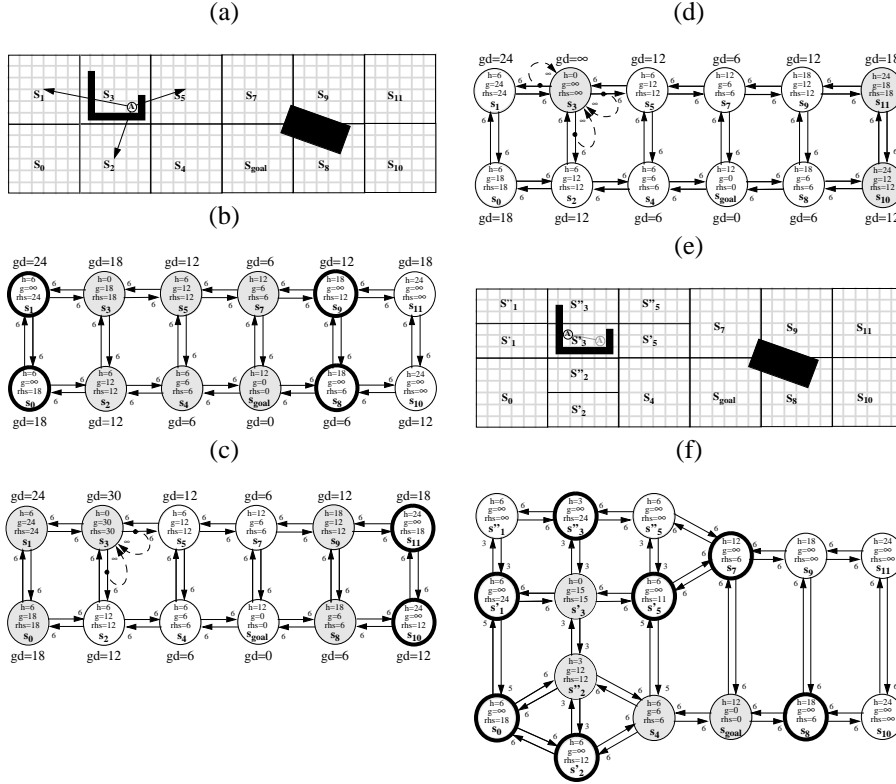

Figure 1: Example behavior of the parti-game algorithm

they are too coarse-grained (for example, because the resolution prevents one from notic- ing small gaps between obstacles) and results in large state spaces that cannot be searched efficiently if they are too fine-grained. The parti-game algorithm solves this dilemma by starting with a coarse discretization and refines it during execution only when and where it is needed (for example, around obstacles), resulting in a nonuniform discretization.

We use a simple two-dimensional robot navigation domain to illustrate the behavior of the parti-game algorithm. Figure 1(a) shows the initial discretization of our example domain into 12 large cells together with the start coordinates of the agent (A) and the goal region (cell containing $s_{goal}$). Thus, it can always attempt to move towards the center of each adjacent cell (that is, cell that its current cell shares a border line with). The agent can initially attempt to move towards the centers of either $s_1$, $s_2$, or $s_5$, as shown in the figure. Figure 1(b) shows the state space that corresponds to the discretized domain under this assumption. Each state corresponds to a cell and each action corresponds to a movement option. The parti-game algorithm initially ignores obstacles and makes the optimistic (and sometimes wrong) assumption that each action deterministically reaches the intended state, for example, that the agent indeed reaches $s_5$ if it is somewhere in $s_3$ and moves towards the center of $s_5$. The costs of an action outcome approximates the Euclidean distance from the center of the old cell of the agent to the center of its new cell.[1] (The cost of the action

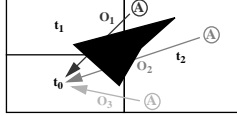

Figure 2: Example of a nondeterministic action

outcome is infinity if the old and new cells are identical since the action then cannot be part of a plan that minimizes the worst-case plan-execution cost from the current state of the agent to $s_{goal}$.) The parti-game algorithm then determines whether the minimax goal distance of the current state $s_{current}$ of the agent is finite. If so, the parti-game algorithm repeatedly chooses the action that minimizes the worst-case plan-execution cost, until the agent reaches $s_{goal}$ or observes additional action outcomes. The minimax goal distance of $s_{current}$ is $gd(s_3) = 18$ and the agent minimizes the worst-case plan-execution cost by moving from $s_3$ towards the centers of either $s_2$ or $s_5$. Assume that it decides to move towards the center of $s_2$. The agent always continues to move until it either gets blocked by an obstacle or enters a new cell. It immediately gets blocked by the obstacle in $s_3$. When the agent observes additional action outcomes it adds them to the state space. Thus, it now assumes that it can end up in either $s_2$ or $s_3$ if it is somewhere in $s_3$ and moves towards the center of $s_2$. The same scenario repeats when the agent first attempts to move towards the center of $s_5$ and then attempts to move towards the center of $s_1$ but gets blocked twice by the obstacle in $s_3$. Figure 1(c) shows the state space after the attempted moves towards the centers of $s_2$ and $s_5$, and Figure 1(d) shows the state space after the attempted move towards the center of $s_1$. The minimax goal distance of $s_{current}$ is now $gd(s_3) = \infty$. We say that $s_{current}$ is unsolvable since an agent in $s_{current}$ is not guaranteed to reach $s_{goal}$ with finite plan-execution cost. In this case, the parti-game algorithm refines the discretization by splitting all solvable cells that border unsolvable cells and all unsolvable cells that border solvable cells. Each cell is split into two cells perpendicular to its longest axis. (The axis of the split is chosen randomly for square cells.) Figure 1(e) shows the new discretization of the domain. The parti-game algorithm then removes those states (and their actions) from the state space that correspond to the old cells and adds states (and actions) for the new cells, again making the optimistic assumption that each action for the new states deterministically reaches the intended state. This ensures that the minimax goal distance of $s_{current}$ becomes finite. Figure 1(f) shows the resulting state space. The parti-game algorithm now repeats the process until either the agent reaches $s_{goal}$ or the domain cannot be discretized any further because the resolution limit is reached.

If all actions either did indeed deterministically reach their intended states or did not change the state of the agent at all (as in the example from Figure 1), then the parti-game algorithm could determine the minimax goal distances of the states with a deterministic search algorithm after it has removed all actions that have an action outcome that leaves the state unchanged (since these actions cannot be part of a plan with minimal worst-case plan-execution cost). However, actions can have additional outcomes, as Figure 2 illustrates. For example, an agent cannot only end up in $t_0$ and $t_2$ but also in $t_1$ if it moves from somewhere in $t_2$ towards the center of $t_0$. The parti-game algorithm therefore needs to determine the minimax goal distances of the states with a minimax search algorithm. Furthermore, the parti-game algorithm repeatedly determines plans that minimize the worst-case plan-

---

that contains the center of the new and old state of the agent. Similarly, the heuristic of a state is computed as the maximum of the absolute differences of the x and y coordinates between the imaginary grid cell that contains the center of the state of the agent and the imaginary grid cell that contains the center of the state in question. Note that the grid is imaginary and never needs to be constructed. Furthermore, it is only used to compute the costs and heuristics and does not restrict either the placement of obstacles or the movement of the agent.

The pseudocode uses the following functions to manage the priority queue $U$: U.Top() returns a state with the smallest priority of all states in $U$. U.TopKey() returns the smallest priority of all states in $U$. (If $U$ is empty, then U.TopKey() returns $[\infty; \infty]$.) U.Pop() deletes the state with the smallest priority in $U$ and returns the state. U.Insert($s$, $k$) inserts $s$ into $U$ with priority $k$. U.Update($s$, $k$) changes the priority of $s$ in $U$ to $k$. (It does nothing if the current priority of $s$ already equals $k$.) Finally, U.Remove($s$) removes $s$ from $U$.

**procedure CalculateKey($s$)**
{01} return $[\min(g(s), rhs(s)) + h(s_{current}, s); \min(g(s), rhs(s))]$;

**procedure Initialize()**
{02} $U = \emptyset$;
{03} for all $s \in S$ $rhs(s) = g(s) = \infty$;
{04} $rhs(s_{goal}) = 0$;
{05} U.Insert($s_{goal}$, CalculateKey($s_{goal}$));

**procedure UpdateState($u$)**
{06} if $(u \neq s_{goal})$ $rhs(u) = \min_{a \in A(u)} \max_{s' \in Succ(u,a)}(c(u,a,s') + g(s'))$;
{07} if $(u \in U)$ U.Remove($u$);
{08} if $(g(u) \neq rhs(u))$ U.Insert($u$, CalculateKey($u$));

**procedure ComputePlan()**
{09} while (U.TopKey() $\dot{<}$ CalculateKey($s_{current}$) OR $rhs(s_{current}) \neq g(s_{current})$)
{10}    $u =$ U.Pop();
{11}    if $(g(u) > rhs(u))$ /* $u$ is locally overconsistent */
{12}        $g(u) = rhs(u)$;
{13}        for all $s \in Pred(u)$ UpdateState($s$);
{14}    else /* $u$ is locally underconsistent */
{15}        $g(u) = \infty$;
{16}        for all $s \in Pred(u) \cup \{u\}$ UpdateState($s$);

**procedure Main()**
{17} $s_{current} = s_{start}$;
{18} Initialize();
{19} ComputePlan();
{20} while $(s_{current} \neq s_{goal})$
{21}    /* if $(rhs(s_{current}) = \infty)$ then the agent is not guaranteed to reach $s_{goal}$ with finite plan-execution cost */
{22}    Execute $\arg \min_{a \in A(s_{current})} \max_{s \in Succ(s_{current},a)}(c(s_{current}, a, s) + g(s))$;
{23}    Set $s_{current}$ to the current state of the agent after the action execution;
{24}    Scan for changed action costs;
{25}    if any action costs have changed
{26}        for all actions with changed action costs $c(u, a, v)$
{27}            Update the action cost $c(u, a, v)$;
{28}            UpdateState($u$);
{29}        for all $s \in U$
{30}            U.Update($s$, CalculateKey($s$));
{31}        ComputePlan();

Figure 3: Minimax LPA*

execution cost from $s_{current}$ to $s_{goal}$. It is therefore important to make the searches fast. In the next sections, we describe Minimax LPA* and how to implement the parti-game algorithm with it. Figures 1(b), (c), (d) and (f) show the state spaces for our example directly after the parti-game algorithm has used Minimax LPA* to determine the minimax goal distance of $s_{current}$. All expanded states (that is, all states whose minimax goal distances have been computed) are shown in gray. Minimax LPA* speeds up the searches by reusing information from previous searches, which is the reason why it expands only three states in Figure 1(d). Minimax LPA* also speeds up the searches by using heuristics to focus them, which is the reason why it expands only four states in Figure 1(f).

# 3   Minimax LPA*

Minimax LPA* repeatedly determines plans that minimize the worst-case plan-execution cost from $s_{current}$ to $s_{goal}$ as the agent moves towards $s_{goal}$ in nondeterministic domains where the costs of actions increase or decrease over time. It generalizes two incremental search algorithms, namely our LPA* [1] and DynamicSWSF-FP [7]. Figure 3 shows the algorithm, that we describe in the following. Numbers in curly braces refer to the line numbers in the figure.

### 3.1 Notation

$S$ denotes the finite set of states. $s_{start} \in S$ is the start state, and $s_{goal} \in S$ is the goal state. $A(s)$ is the set of actions that can be executed in $s \in S$. $Succ(s, a) \subseteq S$ is the set of successor states that can result from the execution of $a \in A(s)$ in $s \in S$. $Succ(s) = \{s' \in S | s' \in Succ(s, a) \text{ for some } a \in A(s)\}$ is the set of successor states of $s \in S$. $Pred(s) = \{s' \in S | s \in Succ(s', a) \text{ for some } a \in A(s')\}$ is the set of predecessor states of $s \in S$. The agent incurs cost $0 < c(s, a, s') \leq \infty$ if the execution of $a \in A(s)$ in $s \in S$ results in $s' \in S$. $0 \leq gd(s) \leq \infty$ is the minimax goal distance of $s \in S$, defined as the solution of the system of equations: $gd(s) = 0$ if $s = s_{goal}$, and $gd(s) = \min_{a \in A(s)} \max_{s' \in Succ(s,a)}(c(s, a, s') + gd(s'))$ for all $s \in S$ with $s \neq s_{goal}$. $s_{current}$ is the current state of the agent, and the minimal worst-case plan-execution cost from $s_{current}$ to $s_{goal}$ is $gd(s_{current})$.

### 3.2 Heuristics and Variables

Minimax LPA* searches backward from $s_{goal}$ to $s_{current}$ and uses heuristics to focus its search. The heuristics need to be non-negative and satisfy $h(s, s) = 0$ and $h(s, s') \leq h(s, s'') + c(s'', a, s')$ for all $s, s', s'' \in S$ and $a \in A(s'')$ with $s' \in Succ(s'', a)$. In other words, the heuristics $h(s, s')$ approximate the best-case plan-execution cost from $s$ to $s'$.

Minimax LPA* maintains two variables for each state that it encounters during the search. The g-value of a state estimates its minimax goal distance. It is carried forward from one search to the next one and can be used after each search to determine a plan that minimizes the worst-case plan-execution cost from $s_{current}$ to $s_{goal}$. The rhs-value of a state also estimates its minimax goal distance. It is a one-step lookahead value based on the g-values of its successors and thus potentially better informed than its g-value. It always satisfies the following relationship (*Invariant 1*): $rhs(s) = 0$ if $s = s_{goal}$, and $rhs(s) = \min_{a \in A(s)} \max_{s' \in Succ(s,a)}(c(s, a, s') + g(s'))$ for all $s \in S$ with $s \neq s_{goal}$. A state is called locally consistent iff its g-value is equal to its rhs-value. Minimax LPA* also maintains a priority queue $U$ that always contains exactly the locally inconsistent states (*Invariant 2*). Their priorities are always identical to their current keys (*Invariant 3*), where the key $k(s)$ of $s$ is the pair $[\min(g(s), rhs(s)) + h(s_{current}, s); \min(g(s), rhs(s))]$, as calculated by CalculateKey(). The keys are compared according to a lexicographic ordering.

### 3.3 Algorithm

Minimax LPA* operates as follows. The main function Main() first calls Initialize() {18} to set the g-values and rhs-values of all states to $\infty$ {03}. The only exception is the rhs-value of $s_{goal}$, that is set to zero {04}. Thus, $s_{goal}$ is the only locally inconsistent state and is inserted into the otherwise empty priority queue {02, 05}. (Note that, in an actual implementation, Minimax LPA* needs to initialize a state only once it encounters it during the search and thus does not need to initialize all states up front. This is important because the number of states can be large and only a few of them might be reached during the search.) Then, Minimax LPA* calls ComputePlan() to compute a plan that minimizes the worst-case plan-execution cost from $s_{current}$ to $s_{goal}$ {19}. If the agent has not reached $s_{goal}$ yet {20}, it executes the first action of the plan {22} and updates $s_{current}$ {23}. It then scans for changed action costs {24}. To maintain Invariants 1, 2, and 3, it calls UpdateState() if some action costs have changed {28} to update the rhs-values and keys of the states potentially affected by the changed action costs as well as their membership in the priority queue if they become locally consistent or inconsistent. It then recalculates the priorities of all states in the priority queue {29-30}. This is necessary because the heuristics change when the agent moves, since they are computed with respect to $s_{current}$. This only

**procedure Main()**
{17'} $s_{current} = s_{start}$;
{18'} while ($s_{current} \neq s_{goal}$)
{19'}    Refine the discretization, if possible (initially: construct the first discretization);
{20'}    Construct the state space that corresponds to the current discretization;
{21'}    Initialize();
{22'}    ComputePlan();
{23'}    if ($rhs(s_{current}) = \infty$) stop with no solution;
{24'}    while ($s_{current} \neq s_{goal}$ AND $rhs(s_{current}) \neq \infty$)
{25'}        $s_{previous} = s_{current}$;
{26'}        Execute $a = \arg\min_{a' \in A(s_{current})} \max_{s' \in Succ(s_{current}, a')}(c(s_{current}, a', s') + g(s'))$;
{27'}        Set $s_{current}$ to the new state of the agent after the action execution;
{28'}        if ($s_{current} \notin Succ(s_{previous}, a)$)
{29'}            $Succ(s_{previous}, a) = Succ(s_{previous}, a) \cup \{s_{current}\}$;
{30'}            $Succ(s_{previous}) = Succ(s_{previous}) \cup \{s_{current}\}$;
{31'}            $Pred(s_{current}) = Pred(s_{current}) \cup \{s_{previous}\}$;
{32'}        UpdateState($s_{previous}$);
{33'}        for all $s \in U$
{34'}            U.Update($s$, CalculateKey($s$));
{35'}        ComputePlan();

Figure 4: Parti-game algorithm using Minimax LPA*

changes the priorities of the states in the priority queue but not which states are locally consistent and thus in the priority queue. Finally, it recalculates a plan {31} and repeats the process.

ComputePlan() operates as follows. It repeatedly removes the locally inconsistent state with the smallest key from the priority queue {10} and expands it {11-16}. It distinguishes two cases. A state is called locally overconsistent iff its g-value is larger than it rhs-value. We can prove that the rhs-value of a locally overconsistent state that is about to be expanded is equal to its minimax goal distance. ComputePlan() therefore sets the g-value of the state to its rhs-value {12}. A state is called locally underconsistent iff its g-value is smaller than it rhs-value. In this case, ComputePlan() sets the g-value of the state to infinity {15}. In either case, ComputePlan() ensures that Invariants 1, 2 and 3 continue to hold {13, 16}. It terminates as soon as $s_{current}$ is locally consistent and its key is less than or equal to the keys of all locally inconsistent states.

**Theorem 1** *ComputePlan() of Minimax LPA\* expands each state at most twice and thus terminates. Assume that, after ComputePlan() terminates, one starts in $s_{current}$ and always executes an action $a \in A(s)$ in the current state $s \in S$ that minimizes $\max_{s' \in Succ(s,a)}(c(s, a, s') + g(s'))$ until $s_{goal}$ is reached (ties can be broken arbitrarily). Then, the plan-execution cost is no larger than the minimax goal distance of $s_{current}$.*

We can also prove several additional theorems about the efficiency of Minimax LPA*, including the fact that it only expands those states whose g-values are not already correct [5]. To reduce its search time, we optimize Minimax LPA* in several ways, for example, to avoid unnecessary re-computations of the rhs-values [5]. We use these optimizations in the experiments. A more detailed description, the intuition behind Minimax LPA*, examples of its operation, and additional theorems and their proofs can be found in [5].

## 4   Using Minimax LPA* to Implement the Parti-Game Algorithm

Figure 4 shows how Minimax LPA* can be used to implement the parti-game algorithm in a more efficient way than with uninformed search from scratch, using some of the functions from Figure 3. Initially, the parti-game algorithm constructs a first (coarse) discretization of the terrain {19'}, constructs the corresponding state space (which includes setting $s_{current}$ to the state of the agent, $s_{goal}$ to the state that includes the goal coordinates, and $Succ(s, a)$, $Succ(s)$, and $Pred(s)$ according to the optimistic assumption that each action deterministically reaches the intended state) {20'}, and uses ComputePlan() to find a first plan from scratch {21'-22'}. If the minimax goal distance of $s_{current}$ is infinity, then it

stops unsuccessfully {23'}. Otherwise, it repeatedly executes the action that minimizes the worst-case plan-execution cost {26'-27'}. If it observes an unknown action outcome {28'}, then it records it {29'-31'}, ensures that Invariants 1, 2 and 3 continue to hold {32'-34'}, uses ComputePlan() to find a new plan incrementally {35'}, and then continues to execute actions until either $s_{current}$ is unsolvable or the agent reaches $s_{goal}$ {24'}. In the former case, it refines the discretization {19'}, uses ComputePlan() to find a new plan from scratch rather than incrementally (because the discretization changes the state space substantially) {20'-22'}, and then repeats the process.

The heuristic of a state in our version of the parti-game algorithm approximates the Euclidean distance from the center of the current cell of the agent to the center of the cell that corresponds to the state in question. The resulting heuristics have the property that we described in Section 3.2. Figures 1(b), (c), (d) and (f) show the heuristics, g-values and rhs-values of all states directly after the call to ComputePlan(). All expanded states are shown in gray, and all locally inconsistent states (that is, states in the priority queue) are shown in bold.

It happens quite frequently that $s_{current}$ is unsolvable and the parti-game algorithm thus has to refine the discretization. If $s_{current}$ is unsolvable, Minimax LPA* expands a large number of states because it has to disprove the existence of a plan rather than find one. We speed up Minimax LPA* for the special case where $s_{current}$ is unsolvable but every other state is solvable since it occurs about half of the time when $s_{current}$ is unsolvable. If states other than $s_{current}$ become unsolvable, some of them need to be predecessors of $s_{current}$. To prove that $s_{current}$ is unsolvable but every other state is solvable, Minimax LPA* can therefore show that all predecessors of $s_{current}$ are solvable but $s_{current}$ itself is not. To show that all predecessors of $s_{current}$ are solvable, Minimax LPA* checks that they are locally consistent, their keys are no larger than U.TopKey(), and their rhs-values are finite. To show that $s_{current}$ is unsolvable, Minimax LPA* checks that the rhs-value of $s_{current}$ is infinite. We use this optimization in the experiments.

## 5 Experimental Results

An implementation of the parti-game algorithm can use search from scratch or incremental search. It can also use uninformed search (using the zero heuristic) and informed search (using the heuristic that we used in the context of the example from Figure 1). We compare the four resulting combinations. All of them use binary heaps to implement the priority queue and the same optimizations but the implementations with search from scratch do not contain any code needed only for incremental search. Since all implementations move the agent in the same way, we compare their number of state expansions, their total run times, and their total search times (that is, the part of the run times spent in the search routines), averaged over 25 two-dimensional terrains of size $1000 \times 1000$ with 30 percent obstacle density, where the resolution limit is one cell. In each case, the goal coordinates are in the center of the terrain, and the start coordinates are in the vertical center and ten percent to the right of the left edge. We also report the average of the ratios of the three measures for each of the four implementations and the one with incremental heuristic search (which is different from the ratio of the averages), together with their 95-percent confidence intervals.

| Implementation of Parti-Game Algorithm with . . . | Expansions | Ratio Expansions | Run Time   (Search Time) | Ratio Run Time (Search Time) |
|---|---|---|---|---|
| Uninformed from Scratch | 69,527,969 | $20.55 \pm 4.12$ | 39 min 51 sec (37 min 43 sec) | $11.83 \pm 3.52$ (15.29 $\pm$ 3.61) |
| Informed from Scratch | 31,303,253 | $8.06 \pm 2.59$ | 22 min 58 sec (20 min 49 sec) | $6.08 \pm 2.50$ ( 7.20 $\pm$ 2.70) |
| Uninformed Incremental | 2,628,879 | $1.23 \pm 0.03$ | 1 min 54 sec ( 1 min 41 sec) | $1.04 \pm 0.02$ ( 1.19 $\pm$ 0.05) |
| Informed Incremental | 2,172,430 | $1.00 \pm 0.00$ | 1 min 45 sec ( 1 min 28 sec) | $1.00 \pm 0.00$ ( 1.00 $\pm$ 0.00) |

The average number of searches, measured by calls to ComputePlan(), is 29,885 until the agent reaches $s_{goal}$. The table shows that the search times of the parti-game algorithm

are substantial due to the large number of searches performed (even though each search is fast), and that the searches take up most of its run time. Thus, speeding up the searches is important. The table also shows that incremental and heuristic search individually speed up the parti-game algorithm and together speed it up even more.

The implementations of the parti-game algorithm in [3] and [6] make slightly different assumptions from ours, for example, minimize state transitions rather than cost. Al-Ansari reports that the original implementation of the parti-game algorithm with value iteration performs about 80 percent and that his implementation with a simple uninformed incremental search method performs about 15 percent of the state expansions of the implementation with uninformed search from scratch [6]. Our results show that our implementation with Minimax LPA* performs about 5 percent of the state expansions of the implementation with uninformed search from scratch. While these results are *not* directly comparable, we have also first results where we ran the original implementation with value iteration and our implementation with Minimax LPA* on a very similar environment and the original implementation expanded one to two orders of magnitude more states than ours even though its number of searches and its final number of states was smaller. However, these results are very preliminary since the time per state expansion is different for the different implementations and it is future work to compare the various implementations of the parti-game algorithm in a common testbed.

## Footnotes

[1] We compute both the costs of action outcomes and the heuristics of states using an imaginary uniform grid, shown in gray in Figures 1(a) and (e), whose cell size corresponds to the resolution limit of the parti-game algorithm. The cost of an action outcome is then computed as the maximum of the absolute values of the differences of the x and y coordinates between the imaginary grid cell

## References

[1] S. Koenig and M. Likhachev. Incremental A*. In T. Dietterich, S. Becker, and Z. Ghahramani, editors, *Advances in Neural Information Processing Systems 14*, Cambridge, MA, 2002. MIT Press.

[2] D. Frigioni, A. Marchetti-Spaccamela, and U. Nanni. Fully dynamic algorithms for maintaining shortest paths trees. *Journal of Algorithms*, 34(2):251–281, 2000.

[3] A. Moore and C. Atkeson. The parti-game algorithm for variable resolution reinforcement learning in multidimensional state-spaces. *Machine Learning*, 21(3):199–233, 1995.

[4] A. Moore and C. Atkeson. Prioritized sweeping: Reinforcement learning with less data and less time. *Machine Learning*, 13(1):103–130, 1993.

[5] M. Likhachev and S. Koenig. Speeding up reinforcement learning with incremental heuristic minimax search. Technical Report GIT-COGSCI-2002/5, College of Computing, Georgia Institute of Technology, Atlanta (Georgia), 2002.

[6] M. Al-Ansari. *Efficient Reinforcement Learning in Continuous Environments*. PhD thesis, College of Computer Science, Northeastern University, Boston (Massachusetts), 2001.

[7] G. Ramalingam and T. Reps. An incremental algorithm for a generalization of the shortest-path problem. *Journal of Algorithms*, 21:267–305, 1996.
